# Learning to parse images of articulated bodies

**Deva Ramanan**
Toyota Technological Institute at Chicago
Chicago, IL 60637
`ramanan@tti-c.org`

## Abstract

We consider the machine vision task of pose estimation from static images, specifically for the case of articulated objects. This problem is hard because of the large number of degrees of freedom to be estimated. Following a established line of research, pose estimation is framed as inference in a probabilistic model. In our experience however, the success of many approaches often lie in the power of the features. Our primary contribution is a novel casting of visual inference as an **iterative parsing** process, where one sequentially learns better and better features tuned to a particular image. We show quantitative results for human pose estimation on a database of over 300 images that suggest our algorithm is competitive with or surpasses the state-of-the-art. Since our procedure is quite general (it does not rely on face or skin detection), we also use it to estimate the poses of horses in the Weizmann database.

## 1 Introduction

We consider the machine vision task of pose estimation from static images, specifically for the case of articulated objects. This problem is hard because of the large number of degrees of freedom to be estimated. Following a established line of research, pose estimation is framed as inference in a probabilistic model. Most approaches tend to focus on algorithms for inference, but in our experience, the low-level image features often dictate success. When reliable features can be extracted (through say, background subtraction or skin detection), approaches tend to do well. This dependence on features tends to be under-emphasized in the literature – one does not want to appear to suffer from "feature-itis". In contrast, we embrace it. Our primary contribution is a novel casting of visual inference as an **iterative parsing** process, where one sequentially learns better and better features tuned to a particular image. Since our approach is fairly general (we do not use any skin or face detectors), we also apply it to estimate horse poses from the Weizmann dataset [1].

Another practical difficulty, specifically with pose estimation, is that of reporting results. It is common for an algorithm to return a set of poses, and the correct one is *manually* selected. This is because the posterior of body poses is often multimodal, a single MAP/mode estimate won't summarize it. Inspired by the language community, we propose a perplexity-based measure for evaluation. We calculate the probability of observing the actual pose under the distribution returned by our algorithm. With such an evaluation procedure, we can quantifiable demonstrate that our approach improves the state-of-the-art.

**Related Work:** Human pose estimation from static images is a very active research area. Most approaches tend to use a people-specific features, such as face/skin/hair detection [6, 4, 12]. Our work relies on the conditional random field (CRF) notion of deformable matching in [9]. Our approach is related to those that simultaneously estimate pose and segment an image [7, 10, 2, 5], since we learn low-level segmentation cues to build part-specific region models. However, we compute no explicit segmentation.

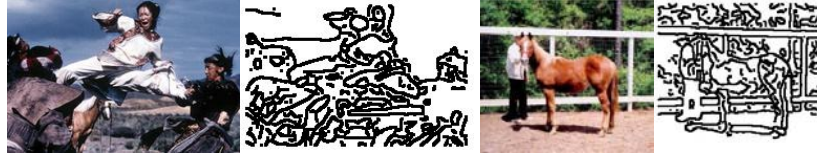

Figure 1: The curse of edges? Edges are attractive because of their invariance – they fire on dark objects in light backgrounds and vice-versa. But without a region model, it can be hard to separate the figure from the background. We describe an iterative algorithm for pose estimation that learns a region model for each body part and for the background. Our algorithm is initialized by the edge maps shown; we show results for these two images in Fig.7 and Fig.8.

## 1.1 Overview

Assume we are given an image of a person, who happens to be a soccer player wearing a white shirt on a green playing field (Fig. 2). We want to estimate the figure's pose. Since we do not know the appearance of the figure or the background, we must use a feature invariant to appearance (Fig.1). We match an **edge**-based deformable model to the image to obtain (soft) estimates of body part positions. In general, we expect these estimates to be poor because the model can be distracted by edges in the background (e.g., the hallucinated leg and the missed arm in Fig. 2). The algorithm uses the estimated body part positions to build a rough region model for each body part and the background – it might learn that the torso is white-ish and the background is green-ish. The algorithm then builds a **region**-based deformable model that looks for white torsos. Soft estimates of body position from the new model are then used to build new region models, and the process is repeated.

As one might suspect, such an iterative procedure is quite sensitive to its starting point – the edge-based deformable model used for initialization and the region-based deformable model used in the first iteration prove crucial. As the iterative procedure is fairly straightforward (Fig.3), most of this paper deals with smart ways of building the deformable models.

## 2 Edge-based deformable model

Our edge-based deformable model is an extension of the one proposed in [9]. The basic probabilistic model is a tree-structured conditional random field (CRF). Let the location of each part $l_i$ be param-

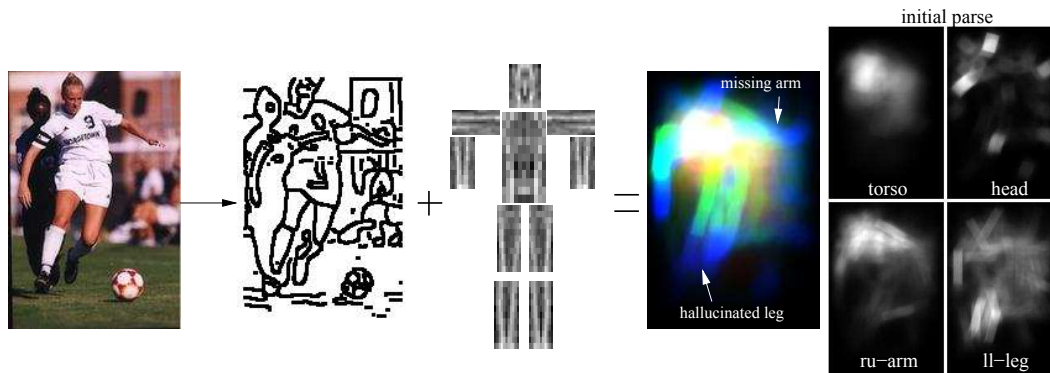

Figure 2: We build a deformable pose model based on edges. Given an image $I$, we use a edge-based deformable model (**middle**) to compute body part locations $P(L|I)$. This defines an initial parse of the image into several body part regions **right**. It is easy to hallucinate extra arms or legs in the negatives spaces between actual body parts (the extra leg). When a body part is surrounded by clutter (the right arm), it is hard to localize. Intuitively, both problems can be solved with low-level segmentation cues. The green region in between the legs is a poor leg candidate because of **figure/ground** cues – it groups better with the background grass. Also, we can find left/right limb pairs by appealing to **symmetry** – if one limb is visible, we can build a model of its appearance, and use it to find the other one. We operationalize both these notions by our iterative parsing procedure in Fig.3.

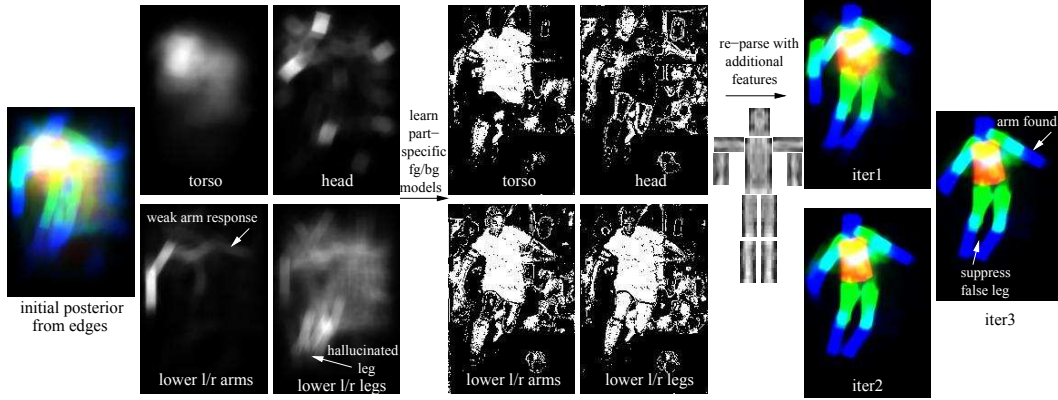

Figure 3: Our iterative parsing procedure. We define a parse to be a soft labeling of pixels into a region type (bg,torso,left lower arm, etc.). We use the initial parse from Fig.2 to build a region model for each part. We learn foreground/background color histogram models. To exploit symmetry in appearance, we learn a single color model for left/right limb pairs. We then label each pixel using the color model (**middle right**). We then use these masks as features for a deformable model that re-computes P($L|I$). This in-turn defines a new parse, and the procedure is repeated.

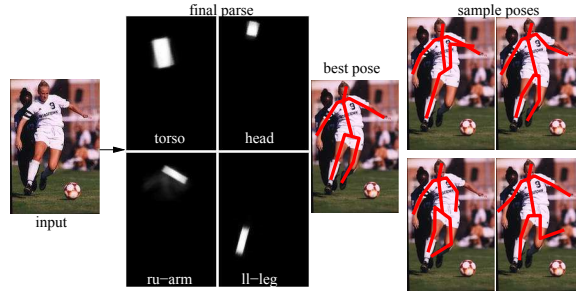

Figure 4: The result of our procedure. Given P($L|I$) from the final iteration, we obtain a clean parse for the image. We can also compute $\hat{L}_{MAP}$ (the most likely pose), and can sample directly from P($L|I$).

eterized by image position and orientation $[x_i, y_i, \theta_i]$. We will assume parts are oriented patches of fixed size, where $(x_i, y_i)$ is the location of the top of the patch. We denote the configuration of a $K$ part model as $L = (l_1 \dots l_K)$.

We can write the deformable model as a log-linear model

$$P(L|I) \propto \exp\left(\sum_{i,j\in E}\psi(l_i - l_j) + \sum_i \phi(l_i)\right) \qquad (1)$$

$\Psi(\mathbf{l_i - l_j})$ corresponds to a spatial prior on the relative arrangement of part $i$ and $j$. For efficient inference, we assume the edge structure $E$ is a tree; each part is connected to at most one parent. Unlike most approaches that assume gaussian shape priors [9, 3], we parameterize our shape model with discrete binning (Fig.5).

$$\psi(l_i - l_i) = \alpha_i^T \text{bin}(l_i - l_j) \qquad (2)$$

Doing so allows us to capture more intricate distributions, at the cost of having more parameters to fit. We write bin($\cdot$) for the vectorized count of spatial and angular histogram bins (a vector of all zeros with a single one for the occupied bin). Here $\alpha_i$ is a model parameter that favors certain (relative) spatial and angular bins for part $i$ with respect to its parent.

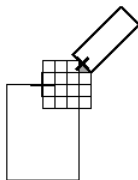

Figure 5: We record the spatial configuration of an arm given the torso by placing a grid on the torso, and noting which bin the arm falls into. We center the grid at the average location of arm in the training data. We likewise bin the angular orientations to define a spatial distribution of arms given torsos.

$\Phi(\mathbf{l_i})$ corresponds to the local image evidence for a part, which we define as

$$\phi(l_i) = \beta_i^T f_i(I(l_i)) \tag{3}$$

We write $f_i(I(l_i))$ for feature vector extracted from the oriented image patch at location $l_i$. In general, $f_i()$ might be part-specific; it could return a binary vector of skin pixels for the the head. In our case, $f_i^e$ returns a binary vector of edges for all parts. We can visualize $\beta_i$ in Fig.6.

**Inference:** The basic machinery we use for inference is message-passing (the sum-product algorithm). Since $E$ is a tree, we first pass "upstream" messages from part $i$ to its parent $j$

We compute the message from part $i$ to $j$ as

$$m_i(l_j) \propto \sum_{l_j} \psi(l_i - l_j) a_i(l_i) \tag{4}$$

$$a_i(l_i) \propto \phi(l_i) \prod_{k \in kids_i} m_k(l_i) \tag{5}$$

Message passing can be performed exhaustively and efficiently with convolutions. If we temporarily ignore orientation and think of $l_i = (x_i, y_i)$, we can represent messages as 2D images. The image $a_i$ is obtained by multiplying together response images from the children of part $i$ and from the imaging model $\phi(l_i)$. $\phi(l_i)$ can be computed by convolving the edge image with the filter $\beta_i$. $m_i(l_j)$ can be computed by convolving $a_i$ with a spatial filter extending over the bins from Fig.5 (with coefficients equal to $\alpha_i$). At the root, the image $a_i$ is the true conditional marginal $P(l_i|I)$. When $l_i$ is 3D, we perform 3D convolutions. We assume $\alpha_i$ is separable so convolutions can be performed separately in each dimension. This means that in practice, computing $\phi(l_i)$ is the computational bottleneck, since that requires convolving the edge image repeatedly with rotated versions of filter $\beta_i$.

Starting from the root, we can pass messages downstream from part $j$ to part $i$ (again with convolutions)

$$P(l_i|I) \propto a_i(l_i) \sum_{l_j} \psi(l_i - l_j) P(l_j|I) \tag{6}$$

For numerical stability, we normalize images to 1 as they are computed. By keeping track of the normalization constants, we can also compute the partition function (which is needed for computing the evaluation score in Sec. 5).

**Learning:** We learn the filters $\alpha_i$ and $\beta_i$ by CRF parameter estimation, as in [9]. We label training images with body part locations $L$, and find the filters that maximize $P(L|I)$ for the training set. This objective function is convex and so we tried various optimization packages, but found simple stochastic gradient ascent to work well. We define the model learned from the edge feature map $f_i^e$ as $\Theta_e = \{\alpha_i^e, \beta_i^e\}$.

## 3   Building a region model

One can use the marginals (for say, the head) to define a soft labeling for the image into head/non-head pixels. One can do this by repeatedly sampling a head location (according to $P(l_i|I)$) and then

rendering a head at the given location and orientation. Let the rendered appearance for part $i$ be an image patch $s_i$; we use a simple rectangular mask. In the limit of infinite samples, one will obtain an image

$$p_i(x, y) = \sum_{x_i, y_i, \theta_i} P(x_i, y_i, \theta_i | I) s_i^{\theta_i}(x - x_i, y - y_i) \tag{7}$$

We call such an image a **parse** for part $i$ (the images on the right from Fig. 2). It is readily computed by convolving $P(l_i | I)$ with rotated versions of patch $s_i$. Given the parse image $p_i$, we learn a color histogram model for part $i$ and "its" background.

$$P(fg_i(k)) \propto \sum_{x,y} p_i(x, y)\delta(im(x, y) = k) \tag{8}$$

$$P(bg_i(k)) \propto \sum_{x,y} (1 - p_i(x, y))\delta(im(x, y) = k) \tag{9}$$

We use the part-specific histogram models to label each pixel as foreground or background with a likelihood ratio test (as shown in Fig.3). To enforce symmetry in appearance, we learn a single color model for left/right limb pairs.

## 4   Region-based deformable model

After an initial parse, our algorithm has built an initial region model for each part (and its background). We use these models to construct binary label images for part $i$: $P(fg_i(im)) > P(bg_i(im))$. We write the oriented patch features extracted from these label images as $f_i^r$ (for "region"-based). We want to use these features to help re-estimate the pose in an image – we using training data to learn how to do so. We learn model parameters for a region-based deformable model $\Theta_r$ by CRF parameter estimation, as in Sec.2.

When learning $\Theta_r$ from training data, defining $f_i^r$ is tricky – should we use the ground-truth part locations to learn the color histogram models? Doing so might be unrealistic – it assumes at "run-time", the edge-based deformable model will always correctly estimate part locations. Rather, we run the edge-based model on the training data, and use the resulting parses to learn the color histogram models. This better mimics the situation at run-time, when we are faced with a new image to parse.

When applying the region-based deformable model, we have already computed the edge responses $\phi^e(l_i) = \beta_i^{eT} f^e(I(l_i))$ (to train the region model). With little additional computational cost, we can add them as an extra feature to the region-based map $f_i^r$. One might think that the region-features eliminate the need for edges – once we know that a person is wearing a white shirt in a green background, why bother with edges? If this was the case, one would learn a zero weight for the edge feature when learning $\beta_i^r$ from training data. We learn roughly *equal* weights for the edge and region features, indicating both cues are complementary rather than redundant.

Given the parse from the region-based model, we can re-learn a color model for each part and the background (and re-parse given the new models, and iterate). In our experience, both the parses and the color models empirically converge after 1-2 iterations (see Fig. 3).

## 5   Results

We have tested our parsing algorithm on two datasets. Most people datasets are quite small, limited to tens of images. We have amassed a dataset of 305 images of people in interesting poses (which will be available on the author's webpage). It has been collected from previous datasets of sports figures and personal pictures. To our knowledge, it is the largest labeled dataset available for human pose recognition. We also have tested our algorithm on the Weizmann dataset of horses [1].

**Evalutation:** Given an image, our parsing procedure returns a distribution over poses $P(L|I)$. Ideally, we want the true pose to have a high probability, and all other poses to have a low value. Given a set of $T$ test images each with a labeled ground-truth pose $\hat{L}_t$, we score performance by computing $-\frac{1}{T}\sum_t \log P(\hat{L}_t | I_t)$. This is equivalent to standard measures of perplexity (up to a log scale) [11].

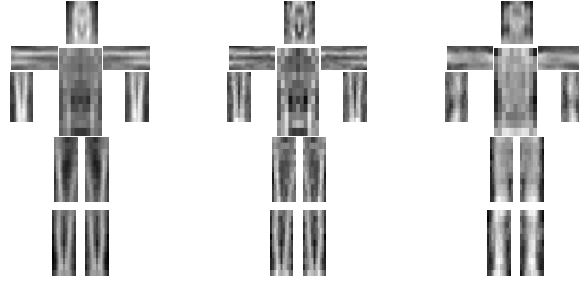

Figure 6: We visualize the part models for our deformable templates – light areas correspond to positive $\beta_i$ weights, and dark corresponds to negative. It is crucial to initialize our iterative procedure with a good edge-based deformable model. Given a collection of training images with labeled body parts, one could build an edge template for each part by averaging (**left**) – this is the standard maximum likelihood (ML) solution. As in [9], we found better results by training $\beta_i^e$ with a conditional random field (CRF) model (**middle**). The CRF edge templates seem to emphasize different features, such as the contours of the head, lower arms, and lower torso. The first re-parsing from Fig.3 is also very crucial – we similarly learn region-based part templates $\beta_i^r$ with a CRF (**right**). These templates focus more on region cues rather than edges. These templates appear more sophisticated than rectangle-based limb detectors [8, 9] – for example, to find upper arms and legs, it seems important to emphasize the edge facing away from the body.

Log-probability of images given model

|  | Iter 0 | Iter 1 | Iter2 |
|---|---|---|---|
| PeopleAll | 62.33 | 55.60 | 57.39 |
| HorsesAll | 51.81 | 47.76 | 45.80 |

Comparison with previous work

|  | Previous | Iter 0 | Iter 1 |
|---|---|---|---|
| USCPeople | 55.85 | 45.77 | 41.49 |

Table 1: Quantitative evaluation. For each image, our parsing procedure returns a distribution of poses. We evaluate our algorithm by looking at a perplexity-based score [11] – the negative log probability of the ground truth pose given the estimated distribution, averaged over the test set. On the **left**, we look at the large datasets of people and horses (each with 300 images). Iter0 corresponds to the distribution computed by the edge-based model, while Iter1 and Iter2 show the results after our iterative parsing with a region-based model. For people, we achieve the best performance after one iteration of the region-based model. For horses, we do better after two iterations. To compare with previous approaches, we look at performance on the 20 image dataset from USC [9, 6]. Compared to [9], our model does better at explaining the ground-truth data.

**People:** We learned a model from the first 100 training images (and their mirror-flipped versions). We learn both $\Theta_e$ and $\Theta_r$ from the same training data. We have evaluated results on the 205 remaining images. We show sample image in Fig.7. We localize some difficult poses quite well, and furthermore, the estimated posterior $P(L|I)$ oftentimes reflects actual ambiguity in the data (ie, if multiple people are present). We quantitatively evaluate results in Table 1. We also compare with a state-of-the-art algorithm from [9], and show better performance on dataset used in that work.

**Horses:** We learn a model from the first 20 training images, and test it on the remaining 280 images. In general, we do quite well. The posterior pose distribution often captures the non-rigid deformations in the body. This suggests we can use the *uncertainty* in our deformable matching algorithm to recover extra information about the object. Looking at the numbers in Table 1, we see that the parses tend do significantly better at capturing the ground-truth poses. We also see that this dataset is easier overall than our set of 305 people poses.

**Discussion:** We have described an iterative parsing approach to pose estimation. Starting with an edge-based detector, we obtain an initial parse and iteratively build better features with which to subsequently parse. We hope this approach of learning image-specific features will prove helpful in other vision tasks.

# References

[1] E. Borenstein and S. Ullman. Class-specific, top-down segmentation. In *ECCV*, 2002.

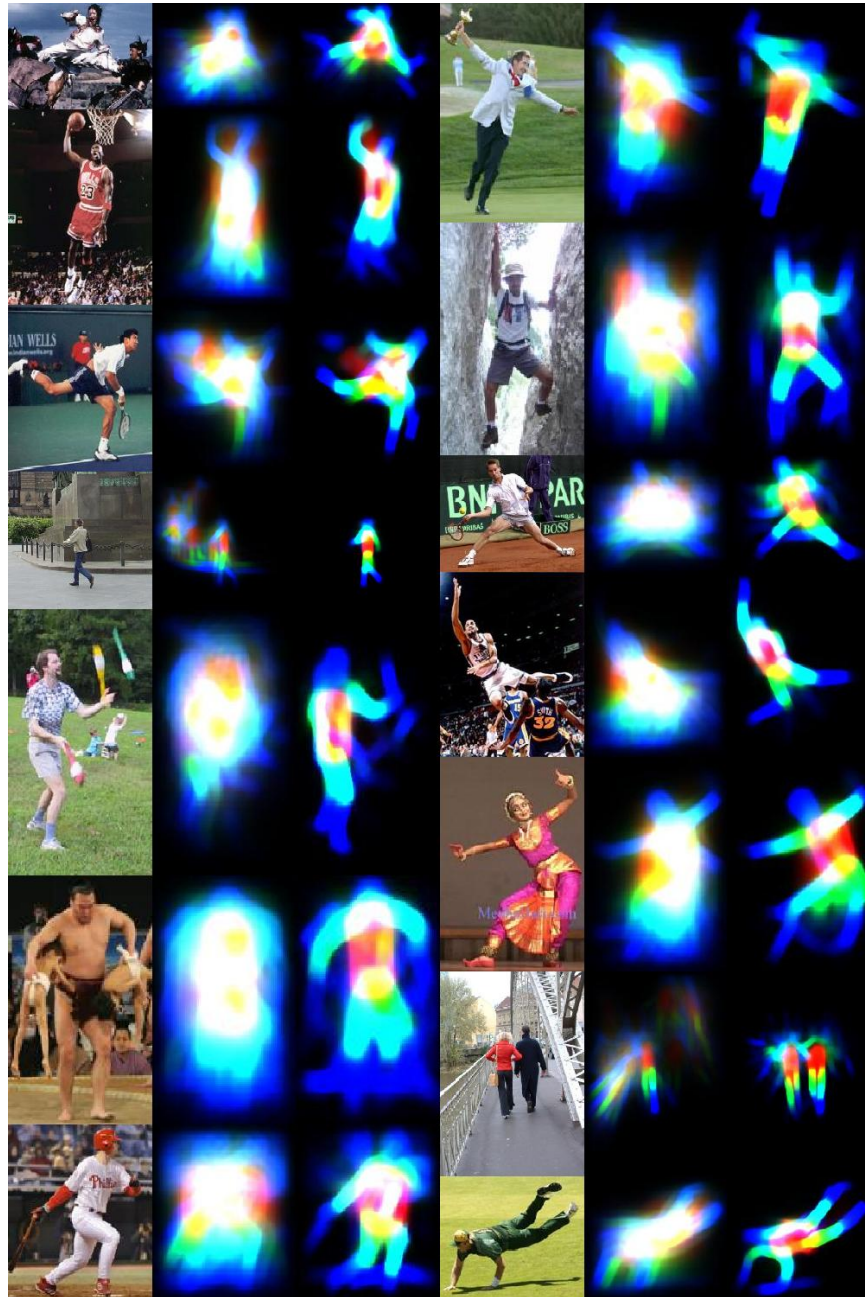

Figure 7: Sample results. We show the original image, the initial edge-based parse, and the final region-based parse. We are able to capture some extreme articulations. In many cases the posterior is ambiguous because the image is (ie, multiple people are present). In particular, it may be surprising that the pair in the bottom-right both are recognized by the region model – this suggests that the the iter-region dissimilarity learned by the color histograms is a much stronger than the foreground similarity. We quantify results in Table 1.

[2] M. Bray, P. Kohli, and P. Torr. Posecut: simultaneous segmentation and 3d pose estimation of humans using dynamic graph-cuts. In *ECCV*, 2006.

[3] P. F. Felzenszwalb and D. P. Huttenlocher. Pictorial structures for object recognition. *Int. J. Computer Vision*, 61(1), January 2005.

[4] M.-H. Y. Gang Hua and Y. Wu. Learning to estimate human pose with data driven belief propagation. In *CVPR*, 2005.

[5] M. Kumar, P. Torr, and A. Zisserman. Objcut. In *CVPR*, 2005.

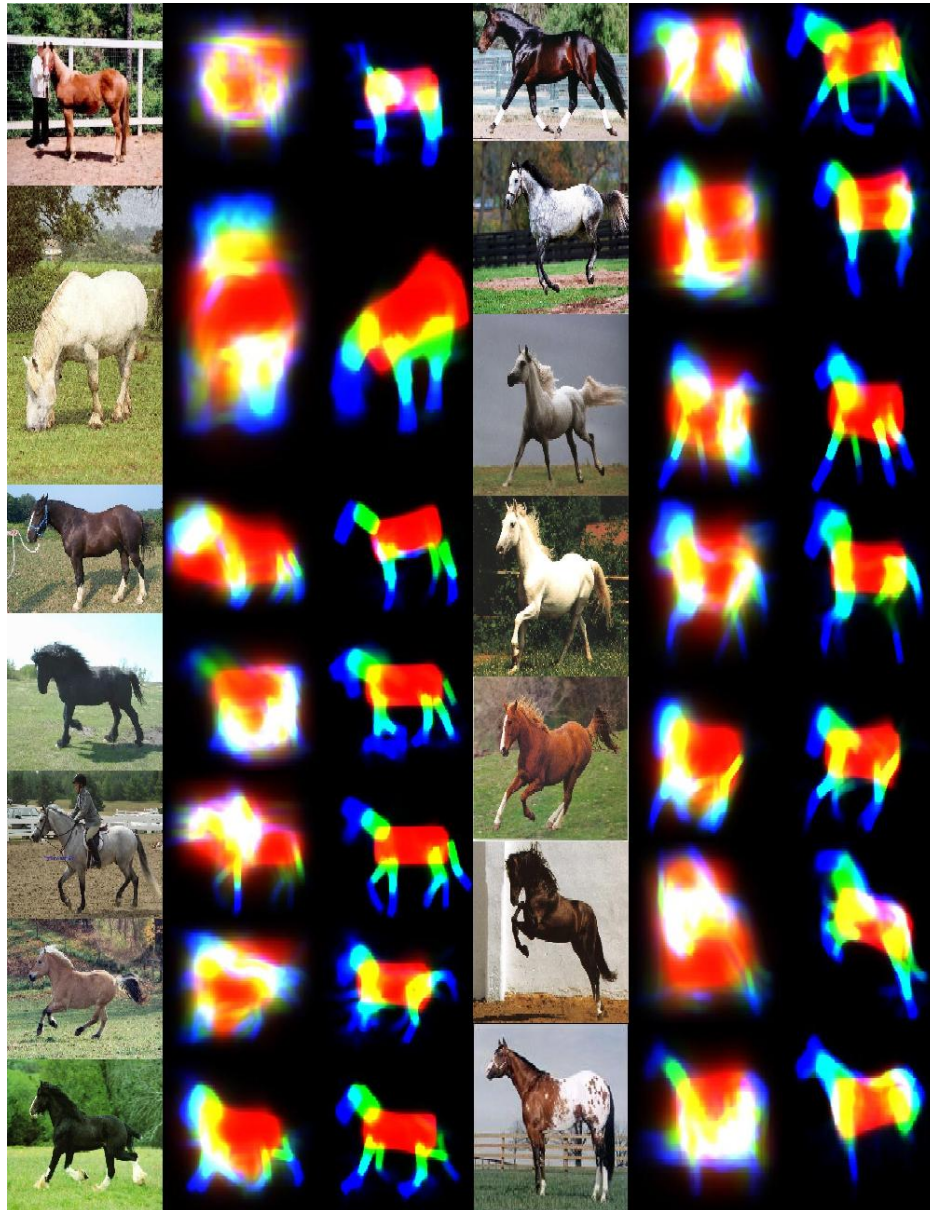

Figure 8: Sample results for horses. Our results tend to be quite good across the entire dataset of 300 images. Even though the horse model is fairly simplistic – a collection of rectangles similar to Fig. 6 – the posterior can capture rich non-rigid deformations of body parts. The Weizmann set of horses seems to be easier than our people dataset - we quantify this with a perplexity score in Table 1.

[6] M. Lee and I. Cohen. Proposal maps driven mcmc for estimating human body pose in static images. In *CVPR*, 2004.

[7] G. Mori, X. Ren, A. Efros, and J. Malik. Recovering human body configurations: Combining segmentation and recognition. In *CVPR*, 2004.

[8] D. Ramanan, D. Forsyth, and A. Zisserman. Strike a pose: Tracking people by finding stylized poses. In *CVPR*, June 2005.

[9] D. Ramanan and C. Sminchisescu. Training deformable models for localization. In *CVPR*, 2006.

[10] X. Ren, A. C. Berg, and J. Malik. Recovering human body configurations using pairwise constraints between parts. In *ICCV*, 2005.

[11] S. Russell and P. Norvig. *Artifical Intelligence: A Modern Approach*, chapter 23, pages 835–836. Prentice Hall, 2nd edition edition, 2003.

[12] J. Zhang, J. Luo, R. Collins, and Y. Liu. Body localization in still images using hierarchical models and hybrid search. In *CVPR*, 2006.
